# Kernel Choice and Classifiability for RKHS Embeddings of Probability Distributions

**Bharath K. Sriperumbudur**
Department of ECE
UC San Diego, La Jolla, USA
bharathsv@ucsd.edu

**Kenji Fukumizu**
The Institute of Statistical Mathematics
Tokyo, Japan
fukumizu@ism.ac.jp

**Arthur Gretton**
Carnegie Mellon University
MPI for Biological Cybernetics
arthur.gretton@gmail.com

**Gert R. G. Lanckriet**
Department of ECE
UC San Diego, La Jolla, USA
gert@ece.ucsd.edu

**Bernhard Schölkopf**
MPI for Biological Cybernetics
Tübingen, Germany
bs@tuebingen.mpg.de

## Abstract

Embeddings of probability measures into reproducing kernel Hilbert spaces have been proposed as a straightforward and practical means of representing and comparing probabilities. In particular, the distance between embeddings (the maximum mean discrepancy, or MMD) has several key advantages over many classical metrics on distributions, namely easy computability, fast convergence and low bias of finite sample estimates. An important requirement of the embedding RKHS is that it be characteristic: in this case, the MMD between two distributions is zero if and only if the distributions coincide. Three new results on the MMD are introduced in the present study. First, it is established that MMD corresponds to the optimal risk of a kernel classifier, thus forming a natural link between the distance between distributions and their ease of classification. An important consequence is that a kernel must be characteristic to guarantee classifiability between distributions in the RKHS. Second, the class of characteristic kernels is broadened to incorporate all *strictly positive definite* kernels: these include non-translation invariant kernels and kernels on non-compact domains. Third, a generalization of the MMD is proposed for families of kernels, as the supremum over MMDs on a class of kernels (for instance the Gaussian kernels with different bandwidths). This extension is necessary to obtain a single distance measure if a large selection or class of characteristic kernels is potentially appropriate. This generalization is reasonable, given that it corresponds to the problem of learning the kernel by minimizing the risk of the corresponding kernel classifier. The generalized MMD is shown to have consistent finite sample estimates, and its performance is demonstrated on a homogeneity testing example.

## 1 Introduction

Kernel methods are broadly established as a useful way of constructing nonlinear algorithms from linear ones, by embedding points into higher dimensional reproducing kernel Hilbert spaces (RKHSs) [9]. A generalization of this idea is to embed probability distributions into RKHSs, giving

us a linear method for dealing with higher order statistics [6, 12, 14]. More specifically, suppose we are given the set $\mathscr{P}$ of all Borel probability measures defined on the topological space $M$, and the RKHS $(\mathcal{H}, k)$ of functions on $M$ with $k$ as its reproducing kernel (r.k.). For $\mathbb{P} \in \mathscr{P}$, denote by $\mathbb{P}k := \int_M k(., x)\, d\mathbb{P}(x)$. If $k$ is measurable and bounded, then we may define the embedding of $\mathbb{P}$ in $\mathcal{H}$ as $\mathbb{P}k \in \mathcal{H}$. The RKHS distance between two such mappings associated with $\mathbb{P}, \mathbb{Q} \in \mathscr{P}$ is called the *maximum mean discrepancy* (MMD) [6, 14], and is written

$$\gamma_k(\mathbb{P}, \mathbb{Q}) = \|\mathbb{P}k - \mathbb{Q}k\|_{\mathcal{H}}. \tag{1}$$

We say that $k$ is *characteristic* [4, 14] if the mapping $\mathbb{P} \mapsto \mathbb{P}k$ is injective, in which case (1) is zero if and only if $\mathbb{P} = \mathbb{Q}$, i.e., $\gamma_k$ is a metric on $\mathscr{P}$. An immediate application of the MMD is to problems of comparing distributions based on finite samples: examples include tests of homogeneity [6], independence [7], and conditional independence [4]. In this application domain, the question of whether $k$ is characteristic is key: without this property, the algorithms can fail through inability to distinguish between particular distributions.

**Characteristic kernels are important in binary classification:** The problem of distinguishing distributions is strongly related to binary classification: indeed, one would expect easily *distinguishable* distributions to be easily *classifiable*.[1] The link between these two problems is especially direct in the case of the MMD: in Section 2, we show that $\gamma_k$ is the negative of the optimal risk (corresponding to a linear loss function) associated with the *Parzen window classifier* [9, 11] (also called *kernel classification rule* [3, Chapter 10]), where the Parzen window turns out to be $k$. We also show that $\gamma_k$ is an upper bound on the margin of a hard-margin support vector machine (SVM). The importance of using characteristic RKHSs is further underlined by this link: if the property does not hold, then there exist distributions that are unclassifiable in the RKHS $\mathcal{H}$. We further strengthen this by showing that characteristic kernels are necessary (and sufficient under certain conditions) to achieve Bayes risk in the kernel-based classification algorithms.

**Characterization of characteristic kernels:** Given the centrality of the characteristic property to both RKHS classification and RKHS distribution testing, we should take particular care in establishing which kernels satisfy this requirement. Early results in this direction include [6], where $k$ is shown to be characteristic on compact $M$ if it is universal in the sense of Steinwart [15, Definition 4]; and [4, 5], which address the case of non-compact $M$, and show that $k$ is characteristic if and only if $\mathcal{H} + \mathbb{R}$ is dense in the Banach space of $p$-power ($p \geq 1$) integrable functions. The conditions in both these studies can be difficult to check and interpret, however, and the restriction of the first to compact $M$ is limiting. In the case of translation invariant kernels, [14] proved the kernel to be characteristic if and only if the support of the Fourier transform of $k$ is the entire $\mathbb{R}^d$, which is a much easier condition to verify. Similar sufficient conditions are obtained by [5] for translation invariant kernels on groups and semi-groups. In Section 3, we expand the class of characteristic kernels to include kernels that may or may not be translation invariant, with the introduction of a novel criterion: *strictly positive definite* kernels (see Definition 3) on $M$ are characteristic.

**Choice of characteristic kernels:** In expanding the families of allowable characteristic kernels, we have so far neglected the question of *which* characteristic kernel to choose. A practitioner asking by how much two samples differ does not want to receive a blizzard of answers for every conceivable kernel and bandwidth setting, but a single measure that satisfies some "reasonable" notion of distance across the family of kernels considered. Thus, in Section 4, we propose a generalization of the MMD, yielding a new distance measure between $\mathbb{P}$ and $\mathbb{Q}$ defined as

$$\gamma(\mathbb{P}, \mathbb{Q}) = \sup\{\gamma_k(\mathbb{P}, \mathbb{Q}) : k \in \mathcal{K}\} = \sup\{\|\mathbb{P}k - \mathbb{Q}k\|_{\mathcal{H}} : k \in \mathcal{K}\}, \tag{2}$$

which is the maximal RKHS distance between $\mathbb{P}$ and $\mathbb{Q}$ over a family, $\mathcal{K}$ of positive definite kernels. For example, $\mathcal{K}$ can be the family of Gaussian kernels on $\mathbb{R}^d$ indexed by the bandwidth parameter. This distance measure is very natural in the light of our results on binary classification (in Section 2): most directly, this corresponds to the problem of learning the kernel by minimizing the risk of the associated Parzen-based classifier. As a less direct justification, we also increase the upper bound on the margin allowed for a hard margin SVM between the samples. To apply the generalized MMD in practice, we must ensure its empirical estimator is consistent. In our main result of Section 4, we provide an empirical estimate of $\gamma(\mathbb{P}, \mathbb{Q})$ based on finite samples, and show that many popular kernels like the Gaussian, Laplacian, and the entire Matérn class on $\mathbb{R}^d$ yield consistent estimates

of $\gamma(\mathbb{P}, \mathbb{Q})$. The proof is based on bounding the *Rademacher chaos complexity* of $\mathcal{K}$, which can be understood as the U-process equivalent of Rademacher complexity [2].

Finally, in Section 5, we provide a simple experimental demonstration that the generalized MMD can be applied in practice to the problem of homogeneity testing. Specifically, we show that when two distributions differ on particular length scales, the kernel selected by the generalized MMD is appropriate to this difference, and the resulting hypothesis test outperforms the heuristic kernel choice employed in earlier studies [6]. The proofs of the results in Sections 2-4 are provided in the supplementary material.

## 2 Characteristic Kernels and Binary Classification

One of the most important applications of the maximum mean discrepancy is in nonparametric hypothesis testing [6, 7, 4], where the characteristic property of $k$ is required to distinguish between probability measures. In the following, we show how MMD naturally appears in binary classification, with reference to the Parzen window classifier and hard-margin SVM. This motivates the need for characteristic $k$ to guarantee that classes arising from different distributions can be classified by kernel-based algorithms.

To this end, let us consider the binary classification problem with $X$ being a $M$-valued random variable, $Y$ being a $\{-1, +1\}$-valued random variable and the product space, $M \times \{-1, +1\}$, being endowed with an induced Borel probability measure $\mu$. A discriminant function, $f$ is a real valued measurable function on $M$, whose sign is used to make a classification decision. Given a loss function $L : \{-1, +1\} \times \mathbb{R} \to \mathbb{R}$, the goal is to choose an $f$ that minimizes the risk associated with $L$, with the optimal $L$-risk being defined as

$$R_{\mathcal{F}_\star}^L = \inf_{f \in \mathcal{F}_\star} \int_M L(y, f(x)) \, d\mu(x, y) = \inf_{f \in \mathcal{F}_\star} \left\{ \varepsilon \int_M L_1(f) \, d\mathbb{P} + (1 - \varepsilon) \int_M L_{-1}(f) \, d\mathbb{Q} \right\}, \quad (3)$$

where $\mathcal{F}_\star$ is the set of all measurable functions on $M$, $L_1(\alpha) := L(1, \alpha)$, $L_{-1}(\alpha) := L(-1, \alpha)$, $\mathbb{P}(X) := \mu(X|Y = +1)$, $\mathbb{Q}(X) := \mu(X|Y = -1)$, $\varepsilon := \mu(M, Y = +1)$. Here, $\mathbb{P}$ and $\mathbb{Q}$ represent the class-conditional distributions and $\varepsilon$ is the prior distribution of class $+1$. Now, we present the result that relates $\gamma_k$ to the optimal risk associated with the Parzen window classifier.

**Theorem 1** ($\gamma_k$ and Parzen classification)**.** *Let $L_1(\alpha) = -\frac{\alpha}{\varepsilon}$ and $L_{-1}(\alpha) = \frac{\alpha}{1-\varepsilon}$. Then, $\gamma_k(\mathbb{P}, \mathbb{Q}) = -R_{\mathcal{F}_k}^L$, where $\mathcal{F}_k = \{f : \|f\|_{\mathcal{H}} \le 1\}$ and $\mathcal{H}$ is an RKHS with a measurable and bounded $k$. Suppose $\{(X_i, Y_i)\}_{i=1}^N$, $X_i \in M$, $Y_i \in \{-1, +1\}$, $\forall i$ is a training sample drawn i.i.d. from $\mu$ and $m = |\{i : Y_i = 1\}|$. If $\widetilde{f} \in \mathcal{F}_k$ is an empirical minimizer of (3) (where $\mathcal{F}_\star$ is replaced by $\mathcal{F}_k$ in (3)), then*

$$sign(\widetilde{f}(x)) = \left\{ \begin{array}{ll} 1, & \frac{1}{m} \sum_{Y_i = 1} k(x, X_i) > \frac{1}{N-m} \sum_{Y_i = -1} k(x, X_i) \\ -1, & \frac{1}{m} \sum_{Y_i = 1} k(x, X_i) \le \frac{1}{N-m} \sum_{Y_i = -1} k(x, X_i) \end{array} \right., \quad (4)$$

*which is the Parzen window classifier.*

Theorem 1 shows that $\gamma_k$ is the negative of the optimal $L$-risk (where $L$ is the linear loss as defined in Theorem 1) associated with the Parzen window classifier. Therefore, if $k$ is not characteristic, which means $\gamma_k(\mathbb{P}, \mathbb{Q}) = 0$ for some $\mathbb{P} \ne \mathbb{Q}$, then $R_{\mathcal{F}_k}^L = 0$, i.e., the risk is maximum (note that since $0 \le \gamma_k(\mathbb{P}, \mathbb{Q}) = -R_{\mathcal{F}_k}^L$, the maximum risk is zero). In other words, if $k$ is characteristic, then the maximum risk is obtained only when $\mathbb{P} = \mathbb{Q}$. This motivates the importance of characteristic kernels in binary classification. In the following, we provide another result which provides a similar motivation for the importance of characteristic kernels in binary classification, wherein we relate $\gamma_k$ to the margin of a hard-margin SVM.

**Theorem 2** ($\gamma_k$ and hard-margin SVM)**.** *Suppose $\{(X_i, Y_i)\}_{i=1}^N$, $X_i \in M$, $Y_i \in \{-1, +1\}$, $\forall i$ is a training sample drawn i.i.d. from $\mu$. Assuming the training sample is separable, let $f_{\text{svm}}$ be the solution to the program, $\inf\{\|f\|_{\mathcal{H}} : Y_i f(X_i) \ge 1, \forall i\}$, where $\mathcal{H}$ is an RKHS with measurable and bounded $k$. If $k$ is characteristic, then*

$$\frac{1}{\|f_{\text{svm}}\|_{\mathcal{H}}} \le \frac{\gamma_k(\mathbb{P}_m, \mathbb{Q}_n)}{2}, \quad (5)$$

*where $\mathbb{P}_m := \frac{1}{m} \sum_{Y_i = 1} \delta_{X_i}$, $\mathbb{Q}_n := \frac{1}{n} \sum_{Y_i = -1} \delta_{X_i}$, $m = |\{i : Y_i = 1\}|$ and $n = N - m$. $\delta_x$ represents the Dirac measure at $x$.*

Theorem 2 provides a bound on the margin of hard-margin SVM in terms of MMD. (5) shows that a smaller MMD between $\mathbb{P}_m$ and $\mathbb{Q}_n$ enforces a smaller margin (i.e., a less smooth classifier, $f_{\text{svm}}$, where smoothness is measured as $\|f_{\text{svm}}\|_{\mathcal{H}}$). We can observe that the bound in (5) may be loose if the number of support vectors is small. Suppose $k$ is not characteristic, then $\gamma_k(\mathbb{P}_m, \mathbb{Q}_n)$ can be zero for $\mathbb{P}_m \neq \mathbb{Q}_n$ and therefore the margin is zero, which means even unlike distributions can become inseparable in this feature representation.

Another justification of using characteristic kernels in kernel-based classification algorithms can be provided by studying the conditions on $\mathcal{H}$ for which the Bayes risk is realized for all $\mu$. Steinwart and Christmann [16, Corollary 5.37] have showed that under certain conditions on $L$, the Bayes risk is achieved for all $\mu$ if and only if $\mathcal{H}$ is dense in $L_p(M, \eta)$ for all $\eta$, where $\eta = \varepsilon\mathbb{P} + (1 - \varepsilon)\mathbb{Q}$. Here, $L_p(M, \eta)$ represents the Banach space of $p$-power integrable functions, where $p \in [1, \infty)$ is dependent on the loss function, $L$. Denseness of $\mathcal{H}$ in $L_p(M, \eta)$ implies $\mathcal{H} + \mathbb{R}$ is dense $L_p(M, \eta)$, which therefore yields that $k$ is characteristic [4, 5]. On the other hand, if constant functions are included in $\mathcal{H}$, then it is easy to show that the characteristic property of $k$ is also sufficient to achieve the Bayes risk. As an example, it can be shown that characteristic kernels are necessary (and sufficient if constant functions are in $\mathcal{H}$) for SVMs to achieve the Bayes risk [16, Example 5.40]. Therefore, the characteristic property of $k$ is fundamental in kernel-based classification algorithms.

Having showed how characteristic kernels play a role in kernel-based classification, in the following section, we provide a novel characterization for them.

## 3 Novel Characterization for Characteristic Kernels

A positive definite (pd) kernel, $k$ is said to be *characteristic* to $\mathscr{P}$ if and only if $\gamma_k(\mathbb{P}, \mathbb{Q}) = 0 \Leftrightarrow \mathbb{P} = \mathbb{Q}, \forall \mathbb{P}, \mathbb{Q} \in \mathscr{P}$. The following result provides a novel characterization for characteristic kernels, which shows that *strictly pd* kernels are characteristic to $\mathscr{P}$. An advantage with this characterization is that it holds for any arbitrary topological space $M$ unlike the earlier characterizations where a group structure on $M$ is assumed [14, 5]. First, we define strictly pd kernels as follows.

**Definition 3** (Strictly positive definite kernels). *Let $M$ be a topological space. A measurable and bounded kernel, $k$ is said to be strictly positive definite if and only if $\int_M \int_M k(x, y) \, d\mu(x) \, d\mu(y) > 0$ for all finite non-zero signed Borel measures, $\mu$ defined on $M$.*

Note that the above definition is *not* equivalent to the usual definition of strictly pd kernels that involves finite sums [16, Definition 4.15]. The above definition is a generalization of *integrally strictly positive definite functions* [17, Section 6]: $\int \int k(x, y) f(x) f(y) \, dx \, dy > 0$ for all $f \in L_2(\mathbb{R}^d)$, which is the strictly positive definiteness of the integral operator given by the kernel. Definition 3 is stronger than the finite sum definition as [16, Theorem 4.62] shows a kernel that is strictly pd in the finite sum sense but not in the integral sense.

**Theorem 4** (Strictly pd kernels are characteristic). *If $k$ is strictly positive definite on $M$, then $k$ is characteristic to $\mathscr{P}$.*

The proof idea is to derive necessary and sufficient conditions for a kernel *not* to be characteristic. We show that choosing $k$ to be strictly pd violates these conditions and $k$ is therefore characteristic to $\mathscr{P}$. Examples of strictly pd kernels on $\mathbb{R}^d$ include $\exp(-\sigma\|x-y\|_2^2)$, $\sigma > 0$, $\exp(-\sigma\|x-y\|_1)$, $\sigma > 0$, $(c^2 + \|x - y\|_2^2)^{-\beta}$, $\beta > 0$, $c > 0$, $B_{2l+1}$-splines etc. Note that $\tilde{k}(x, y) = f(x)k(x, y)f(y)$ is a strictly pd kernel if $k$ is strictly pd, where $f : M \to \mathbb{R}$ is a bounded continuous function. Therefore, *translation-variant* strictly pd kernels can be obtained by choosing $k$ to be a translation invariant strictly pd kernel. A simple example of a *translation-variant* kernel that is a strictly pd kernel on compact sets of $\mathbb{R}^d$ is $\tilde{k}(x, y) = \exp(\sigma x^T y)$, $\sigma > 0$, where we have chosen $f(.) = \exp(\sigma\|.\|_2^2/2)$ and $k(x, y) = \exp(-\sigma\|x - y\|_2^2/2)$, $\sigma > 0$. Therefore, $\tilde{k}$ is characteristic on compact sets of $\mathbb{R}^d$, which is the same result that follows from the universality of $\tilde{k}$ [15, Section 3, Example 1].

The following result in [10], which is based on the usual definition of strictly pd kernels, can be obtained as a corollary to Theorem 4.

**Corollary 5** ([10]). *Let $X = \{x_i\}_{i=1}^m \subset M$, $Y = \{y_j\}_{j=1}^n \subset M$ and assume that $x_i \neq x_j$, $y_i \neq y_j$, $\forall i, j$. Suppose $k$ is strictly positive definite. Then $\sum_{i=1}^m \alpha_i k(., x_i) = \sum_{j=1}^n \beta_j k(., y_j)$ for some $\alpha_i, \beta_j \in \mathbb{R}\backslash\{0\} \Rightarrow X = Y$.*

Suppose we choose $\alpha_i = \frac{1}{m}, \forall i$ and $\beta_j = \frac{1}{n}, \forall j$ in Corollary 5. Then $\sum_{i=1}^m \alpha_i k(., x_i)$ and $\sum_{j=1}^n \beta_j k(., y_j)$ represent the mean functions in $\mathcal{H}$. Note that the Parzen classifier in (4)

is a mean classifier (that separates the mean functions) in $\mathcal{H}$, i.e., $\text{sign}(\langle k(.,x), w \rangle_{\mathcal{H}})$, where $w = \frac{1}{m}\sum_{i=1}^{m} k(.,x_i) - \frac{1}{n}\sum_{i=1}^{n} k(.,y_i)$. Suppose $k$ is strictly pd (more generally, suppose $k$ is characteristic). Then, by Corollary 5, the normal vector, $w$ to the hyperplane in $\mathcal{H}$ passing through the origin is zero, i.e., the mean functions coincide (and are therefore not classifiable) if and only if $X = Y$.

## 4 Generalizing the MMD for Classes of Characteristic Kernels

The discussion so far has been related to the characteristic property of $k$ that makes $\gamma_k$ a metric on $\mathscr{P}$. We have seen that this characteristic property is of prime importance both in distribution testing, and to ensure classifiability of dissimilar distributions in the RKHS. We have not yet addressed how to choose among a selection/family of characteristic kernels, given a particular pair of distributions we wish to discriminate between. We introduce one approach to this problem in the present section.

Let $M = \mathbb{R}^d$ and $k_\sigma(x,y) = \exp(-\sigma\|x-y\|_2^2)$, $\sigma \in \mathbb{R}_+$, where $\sigma$ represents the bandwidth parameter. $\{k_\sigma : \sigma \in \mathbb{R}_+\}$ is the family of Gaussian kernels and $\{\gamma_{k_\sigma} : \sigma \in \mathbb{R}_+\}$ is the family of MMDs indexed by the kernel parameter, $\sigma$. Note that $k_\sigma$ is characteristic for any $\sigma \in \mathbb{R}_{++}$ and therefore $\gamma_{k_\sigma}$ is a metric on $\mathscr{P}$ for any $\sigma \in \mathbb{R}_{++}$. However, in practice, one would prefer a single number that defines the distance between $\mathbb{P}$ and $\mathbb{Q}$. The question therefore to be addressed is how to choose appropriate $\sigma$. The choice of $\sigma$ has important implications on the statistical aspect of $\gamma_{k_\sigma}$. Note that as $\sigma \to 0$, $k_\sigma \to 1$ and as $\sigma \to \infty$, $k_\sigma \to 0$ a.e., which means $\gamma_{k_\sigma}(\mathbb{P},\mathbb{Q}) \to 0$ as $\sigma \to 0$ or $\sigma \to \infty$ for all $\mathbb{P},\mathbb{Q} \in \mathscr{P}$ (this behavior is also exhibited by $k_\sigma(x,y) = \exp(-\sigma\|x-y\|_1)$ and $k_\sigma(x,y) = \sigma^2/(\sigma^2 + \|x-y\|_2^2)$, which are also characteristic). This means choosing *sufficiently small* or *sufficiently large* $\sigma$ (depending on $\mathbb{P}$ and $\mathbb{Q}$) makes $\gamma_{k_\sigma}(\mathbb{P},\mathbb{Q})$ arbitrarily small. Therefore, $\sigma$ has to be chosen appropriately in applications to effectively distinguish between $\mathbb{P}$ and $\mathbb{Q}$. Presently, the applications involving MMD set $\sigma$ heuristically [6, 7].

To generalize the MMD to families of kernels, we propose the following modification to $\gamma_k$, which yields a pseudometric on $\mathscr{P}$,

$$\gamma(\mathbb{P},\mathbb{Q}) = \sup\{\gamma_k(\mathbb{P},\mathbb{Q}) : k \in \mathcal{K}\} = \sup\{\|\mathbb{P}k - \mathbb{Q}k\|_{\mathcal{H}} : k \in \mathcal{K}\}. \tag{6}$$

Note that $\gamma$ is the maximal RKHS distance between $\mathbb{P}$ and $\mathbb{Q}$ over a family, $\mathcal{K}$ of positive definite kernels. It is easy to check that if any $k \in \mathcal{K}$ is characteristic, then $\gamma$ is a metric on $\mathscr{P}$. Examples for $\mathcal{K}$ include: $\mathcal{K}_g := \{e^{-\sigma\|x-y\|_2^2}, x,y \in \mathbb{R}^d : \sigma \in \mathbb{R}_+\}$; $\mathcal{K}_l := \{e^{-\sigma\|x-y\|_1}, x,y \in \mathbb{R}^d : \sigma \in \mathbb{R}_+\}$; $\mathcal{K}_\psi := \{e^{-\sigma\psi(x,y)}, x,y \in M : \sigma \in \mathbb{R}_+\}$, where $\psi : M \times M \to \mathbb{R}$ is a negative definite kernel; $\mathcal{K}_{rbf} := \{\int_0^\infty e^{-\lambda\|x-y\|_2^2}\,d\mu_\sigma(\lambda), x,y \in \mathbb{R}^d, \mu_\sigma \in \mathscr{M}^+ : \sigma \in \Sigma \subset \mathbb{R}^d\}$, where $\mathscr{M}^+$ is the set of all finite nonnegative Borel measures, $\mu_\sigma$ on $\mathbb{R}_+$ that are not concentrated at zero, etc.

The proposal of $\gamma(\mathbb{P},\mathbb{Q})$ in (6) can be motivated by the connection that we have established in Section 2 between $\gamma_k$ and the Parzen window classifier. Since the Parzen window classifier depends on the kernel, $k$, one can propose to learn the kernel like in support vector machines [8], wherein the kernel is chosen such that $R_{\mathcal{F}_k}^L$ in Theorem 1 is minimized over $k \in \mathcal{K}$, i.e., $\inf_{k \in \mathcal{K}} R_{\mathcal{F}_k}^L = -\sup_{k \in \mathcal{K}} \gamma_k(\mathbb{P},\mathbb{Q}) = -\gamma(\mathbb{P},\mathbb{Q})$. A similar motivation for $\gamma$ can be provided based on (5) as learning the kernel in a hard-margin SVM by maximizing its margin.

At this point, we briefly discuss the issue of normalized vs. unnormalized kernel families, $\mathcal{K}$ in (6). We say a translation-invariant kernel, $k$ on $\mathbb{R}^d$ is normalized if $\int_M \psi(y)\,dy = c$ (some positive constant independent of the kernel parameter), where $k(x,y) = \psi(x-y)$. $\mathcal{K}$ is a normalized kernel family if every kernel in $\mathcal{K}$ is normalized. If $\mathcal{K}$ is not normalized, we say it is unnormalized. For example, it is easy to see that $\mathcal{K}_g$ and $\mathcal{K}_l$ are unnormalized kernel families. Let us consider the normalized Gaussian family, $\mathcal{K}_g^n = \{(\sigma/\pi)^{d/2}e^{-\sigma\|x-y\|_2^2}, x,y \in \mathbb{R}^d : \sigma \in [\sigma_0,\infty)\}$. It can be shown that for any $k_\sigma, k_\tau \in \mathcal{K}_g^n$, $0 < \sigma < \tau < \infty$, we have $\gamma_{k_\sigma}(\mathbb{P},\mathbb{Q}) \geq \gamma_{k_\tau}(\mathbb{P},\mathbb{Q})$, which means, $\gamma(\mathbb{P},\mathbb{Q}) = \gamma_{\sigma_0}(\mathbb{P},\mathbb{Q})$. Therefore, the generalized MMD reduces to a single kernel MMD. A similar result also holds for the normalized inverse-quadratic kernel family, $\{\sqrt{2\sigma^2/\pi}(\sigma^2 + \|x-y\|_2^2)^{-1}, x,y \in \mathbb{R} : \sigma \in [\sigma_0,\infty)\}$. These examples show that the generalized MMD definition is usually not very useful if $\mathcal{K}$ is a normalized kernel family. In addition, $\sigma_0$ should be chosen beforehand, which is equivalent to heuristically setting the kernel parameter in $\gamma_k$. Note that $\sigma_0$ cannot be zero because in the limiting case of $\sigma \to 0$, the kernels approach a Dirac distribution, which means the limiting kernel is not bounded and therefore the definition of MMD in (1) does not hold. So, in this work, we consider unnormalized kernel families to render the definition of generalized MMD in (6) useful.

To use $\gamma$ in statistical applications where $\mathbb{P}$ and $\mathbb{Q}$ are known only through i.i.d. samples $\{X_i\}_{i=1}^m$ and $\{Y_i\}_{i=1}^n$ respectively, we require its estimator $\gamma(\mathbb{P}_m, \mathbb{Q}_n)$ to be consistent, where $\mathbb{P}_m$ and $\mathbb{Q}_n$ represent the empirical measures based on $\{X_i\}_{i=1}^m$ and $\{Y_j\}_{j=1}^n$. For $k$ measurable and bounded, [6, 12] have shown that $\gamma_k(\mathbb{P}_m, \mathbb{Q}_n)$ is a $\sqrt{mn/(m+n)}$-consistent estimator of $\gamma_k(\mathbb{P}, \mathbb{Q})$. The statistical consistency of $\gamma(\mathbb{P}_m, \mathbb{Q}_n)$ is established in the following theorem, which uses tools from *U-process theory* [2, Chapters 3,5]. We begin with the following definition.

**Definition 6** (Rademacher chaos). *Let $\mathcal{G}$ be a class of functions on $M \times M$ and $\{\rho_i\}_{i=1}^n$ be independent Rademacher random variables, i.e., $\Pr(\rho_i = 1) = \Pr(\rho_i = -1) = \frac{1}{2}$. The homogeneous Rademacher chaos process of order two with respect to $\{\rho_i\}_{i=1}^n$ is defined as $\{n^{-1}\sum_{i<j}^n \rho_i \rho_j g(x_i, x_j) : g \in \mathcal{G}\}$ for some $\{x_i\}_{i=1}^n \subset M$. The Rademacher chaos complexity over $\mathcal{G}$ is defined as*

$$U_n(\mathcal{G}; \{x_i\}_{i=1}^n) := \mathbb{E}_\rho \sup_{g \in \mathcal{G}} \left| \frac{1}{n} \sum_{i<j}^n \rho_i \rho_j g(x_i, x_j) \right|. \tag{7}$$

We now provide the main result of the present section.

**Theorem 7** (Consistency of $\gamma(\mathbb{P}_m, \mathbb{Q}_n)$). *Let every $k \in \mathcal{K}$ be measurable and bounded with $\nu := \sup_{k \in \mathcal{K}, x \in M} k(x,x) < \infty$. Then, with probability at least $1 - \delta$, $|\gamma(\mathbb{P}_m, \mathbb{Q}_n) - \gamma(\mathbb{P}, \mathbb{Q})| \leq A$, where*

$$A = \sqrt{\frac{16 U_m(\mathcal{K}; \{X_i\})}{m} + \frac{16 U_n(\mathcal{K}; \{Y_i\})}{n}} + \frac{(\sqrt{8\nu} + \sqrt{36\nu \log \frac{4}{\delta}})\sqrt{m+n}}{\sqrt{mn}}. \tag{8}$$

From (8), it is clear that if $U_m(\mathcal{K}; \{X_i\}) = O_{\mathbb{P}}(1)$ and $U_n(\mathcal{K}; \{Y_i\}) = O_{\mathbb{Q}}(1)$, then $\gamma(\mathbb{P}_m, \mathbb{Q}_n) \overset{a.s.}{\to} \gamma(\mathbb{P}, \mathbb{Q})$. The following result provides a bound on $U_m(\mathcal{K}; \{X_i\})$ in terms of the entropy integral.

**Lemma 8** (Entropy bound). *For any $\mathcal{K}$ as in Theorem 7 with $0 \in \mathcal{K}$, there exists a universal constant $C$ such that*

$$U_m(\mathcal{K}; \{X_i\}_{i=1}^m) \leq C \int_0^\nu \log \mathcal{N}(\mathcal{K}, D, \epsilon)\, d\epsilon, \tag{9}$$

*where $D(k_1, k_2) = \frac{1}{m} \left[ \sum_{i<j}^m (k_1(X_i, X_j) - k_2(X_i, X_j))^2 \right]^{\frac{1}{2}}$. $\mathcal{N}(\mathcal{K}, D, \epsilon)$ represents the $\epsilon$-covering number of $\mathcal{K}$ with respect to the metric $D$.*

Assuming $\mathcal{K}$ to be a VC-subgraph class, the following result, as a corollary to Lemma 8 provides an estimate of $U_m(\mathcal{K}; \{X_i\}_{i=1}^m)$. Before presenting the result, we first provide the definition of a VC-subgraph class.

**Definition 9** (VC-subgraph class). *The subgraph of a function $g : M \times \mathbb{R}$ is the subset of $M \times \mathbb{R}$ given by $\{(x, t) : t < g(x)\}$. A collection $\mathcal{G}$ of measurable functions on a sample space is called a VC-subgraph class, if the collection of all subgraphs of the functions in $\mathcal{G}$ forms a VC-class of sets (in $M \times \mathbb{R}$).*

The VC-index (also called the VC-dimension) of a VC-subgraph class, $\mathcal{G}$ is the same as the *pseudo-dimension* of $\mathcal{G}$. See [1, Definition 11.1] for details.

**Corollary 10** ($U_m(\mathcal{K}; \{X_i\})$ for VC-subgraph, $\mathcal{K}$). *Suppose $\mathcal{K}$ is a VC-subgraph class with $V(\mathcal{K})$ being the VC-index. Assume $\mathcal{K}$ satisfies the conditions in Theorem 7 and $0 \in \mathcal{K}$. Then*

$$U_m(\mathcal{K}; \{X_i\}) \leq C\nu \log(C_1 V(\mathcal{K})(16e^9)^{V(\mathcal{K})}), \tag{10}$$

*for some universal constants $C$ and $C_1$.*

Using (10) in (8), we have $|\gamma(\mathbb{P}_m, \mathbb{Q}_n) - \gamma(\mathbb{P}, \mathbb{Q})| = O_{\mathbb{P}, \mathbb{Q}}(\sqrt{(m+n)/mn})$ and by the Borel-Cantelli lemma, $|\gamma(\mathbb{P}_m, \mathbb{Q}_n) - \gamma(\mathbb{P}, \mathbb{Q})| \overset{a.s.}{\to} 0$. Now, the question reduces to which of the kernel classes, $\mathcal{K}$ have $V(\mathcal{K}) < \infty$. [18, Lemma 12] showed that $V(\mathcal{K}_g) = 1$ (also see [19]) and $U_m(\mathcal{K}_{rbf}) \leq C_2 U_m(\mathcal{K}_g)$, where $C_2 < \infty$. It can be shown that $V(\mathcal{K}_\psi) = 1$ and $V(\mathcal{K}_l) = 1$. All these classes satisfy the conditions of Theorem 7 and Corollary 10 and therefore provide consistent estimates of $\gamma(\mathbb{P}, \mathbb{Q})$ for any $\mathbb{P}, \mathbb{Q} \in \mathscr{P}$. Examples of kernels on $\mathbb{R}^d$ that are covered by these classes include the Gaussian, Laplacian, inverse multiquadrics, Matérn class etc. Other choices for $\mathcal{K}$ that are popular in machine learning are the linear combination of kernels, $\mathcal{K}_{\text{lin}} := \{k_\lambda = \sum_{i=1}^l \lambda_i k_i \,|\, k_\lambda \text{ is pd}, \sum_{i=1}^l \lambda_i = 1\}$ and $\mathcal{K}_{\text{con}} := \{k_\lambda = \sum_{i=1}^l \lambda_i k_i \,|\, \lambda_i \geq 0, \sum_{i=1}^l \lambda_i = 1\}$. [13, Lemma 7] have shown that $V(\mathcal{K}_{\text{con}}) \leq V(\mathcal{K}_{\text{lin}}) \leq l$. Therefore, instead of using a class based on a fixed, parameterized kernel, one can also use a finite linear combination of kernels to compute $\gamma$.

So far, we have presented the metric property and statistical consistency (of the empirical estimator) of $\gamma$. Now, the question is how do we compute $\gamma(\mathbb{P}_m, \mathbb{Q}_n)$ in practice. To show this, in the following, we present two examples.

**Example 11.** *Suppose $\mathcal{K} = \mathcal{K}_g$. Then, $\gamma(\mathbb{P}_m, \mathbb{Q}_n)$ can be written as*

$$\gamma^2(\mathbb{P}_m, \mathbb{Q}_n) = \sup_{\sigma \in \mathbb{R}_+} \left[ \sum_{i,j=1}^{m} \frac{e^{-\sigma\|X_i - X_j\|^2}}{m^2} + \sum_{i,j=1}^{n} \frac{e^{-\sigma\|Y_i - Y_j\|^2}}{n^2} - 2 \sum_{i,j=1}^{m,n} \frac{e^{-\sigma\|X_i - Y_j\|^2}}{mn} \right]. \quad (11)$$

*The optimum $\sigma^*$ can be obtained by solving (11) and $\gamma(\mathbb{P}_m, \mathbb{Q}_n) = \|\mathbb{P}_m k_{\sigma^*} - \mathbb{Q}_n k_{\sigma^*}\|_{\mathcal{H}_{\sigma^\star}}$.*

**Example 12.** *Suppose $\mathcal{K} = \mathcal{K}_{con}$. Then, $\gamma(\mathbb{P}_m, \mathbb{Q}_n)$ becomes*

$$
\begin{aligned}
\gamma^2(\mathbb{P}_m, \mathbb{Q}_n) &= \sup_{k \in \mathcal{K}_{con}} \|\mathbb{P}_m k - \mathbb{Q}_n k\|_{\mathcal{H}}^2 = \sup_{k \in \mathcal{K}_{con}} \int \int k \, d(\mathbb{P}_m - \mathbb{Q}_n) \otimes (\mathbb{P}_m - \mathbb{Q}_n) \\
&= \sup\{\boldsymbol{\lambda}^T \boldsymbol{a} : \boldsymbol{\lambda}^T \mathbf{1} = 1, \, \boldsymbol{\lambda} \succeq 0\}, \quad (12)
\end{aligned}
$$

*where we have replaced $k$ by $\sum_{i=1}^{l} \lambda_i k_i$. Here $\boldsymbol{\lambda} = (\lambda_1, \dots, \lambda_l)$ and $(\boldsymbol{a})_i = \|\mathbb{P}_m k_i - \mathbb{Q}_n k_i\|_{\mathcal{H}_i}^2 = \frac{1}{m^2} \sum_{a,b=1}^{m} k_i(X_a, X_b) + \frac{1}{n^2} \sum_{a,b=1}^{n} k_i(Y_a, Y_b) - \frac{2}{mn} \sum_{a,b=1}^{m,n} k_i(X_a, Y_b)$. It is easy to see that $\gamma^2(\mathbb{P}_m, \mathbb{Q}_n) = \max_{1 \le i \le l}(\boldsymbol{a})_i$.*

Similar examples can be provided for other $\mathcal{K}$, where $\gamma(\mathbb{P}_m, \mathbb{Q}_n)$ can be computed by solving a semidefinite program ($\mathcal{K} = \mathcal{K}_{\text{lin}}$) or by the constrained gradient descent ( $\mathcal{K} = \mathcal{K}_l, \mathcal{K}_{rbf}$).

Finally, while the approach in (6) to generalizing $\gamma_k$ is our focus in this paper, an alternative Bayesian strategy would be to define a non-negative finite measure $\lambda$ over $\mathcal{K}$, and to average $\gamma_k$ over that measure, i.e., $\beta(\mathbb{P}, \mathbb{Q}) := \int_{\mathcal{K}} \gamma_k(\mathbb{P}, \mathbb{Q}) \, d\lambda(k)$. This also yields a pseudometric on $\mathscr{P}$. That said, $\beta(\mathbb{P}, \mathbb{Q}) \le \lambda(\mathcal{K})\gamma(\mathbb{P}, \mathbb{Q}), \forall \mathbb{P}, \mathbb{Q}$, which means if $\mathbb{P}$ and $\mathbb{Q}$ can be distinguished by $\beta$, they can be distinguished by $\gamma$, but not vice-versa. In this sense, $\gamma$ is stronger than $\beta$. One further complication with the Bayesian approach is in defining a sensible $\lambda$ over $\mathcal{K}$. Note that $\gamma_{k_0}$ (single kernel MMD based on $k_0$) can be obtained by defining $\lambda(k) = \delta(k - k_0)$ in $\beta(\mathbb{P}, \mathbb{Q})$.

## 5 Experiments

In this section, we present a benchmark experiment that illustrates the generalized MMD proposed in Section 4 is preferred above the single kernel MMD where the kernel parameter is set heuristically. The experimental setup is as follows.

Let $p = N(0, \sigma_p^2)$, a normal distribution in $\mathbb{R}$ with zero mean and variance, $\sigma_p^2$. Let $q$ be the perturbed version of $p$, given as $q(x) = p(x)(1 + \sin \nu x)$. Here $p$ and $q$ are the densities associated with $\mathbb{P}$ and $\mathbb{Q}$ respectively. It is easy to see that $q$ differs from $p$ at increasing frequencies with increasing $\nu$. Let $k(x, y) = \exp(-(x - y)^2/\sigma)$. Now, the goal is that given random samples drawn i.i.d. from $\mathbb{P}$ and $\mathbb{Q}$ (with $\nu$ fixed), we would like to test $H_0 : \mathbb{P} = \mathbb{Q}$ vs. $H_1 : \mathbb{P} \neq \mathbb{Q}$. The idea is that as $\nu$ increases, it will be harder to distinguish between $\mathbb{P}$ and $\mathbb{Q}$ for a fixed sample size. Therefore, using this setup we can verify whether the adaptive bandwidth selection achieved by $\gamma$ (as the test statistic) helps to distinguish between $\mathbb{P}$ and $\mathbb{Q}$ at higher $\nu$ compared to $\gamma_k$ with a heuristic $\sigma$. To this end, using $\gamma(\mathbb{P}_m, \mathbb{Q}_n)$ and $\gamma_k(\mathbb{P}_m, \mathbb{Q}_n)$ (with various $\sigma$) as test statistics $T_{mn}$, we design a test that returns $H_0$ if $T_{mn} \le c_{mn}$, and $H_1$ otherwise. The problem therefore reduces to finding $c_{mn}$. $c_{mn}$ is determined as the $(1 - \alpha)$ quantile of the asymptotic distribution of $T_{mn}$ under $H_0$, which therefore fixes the type-I error (the probability of rejecting $H_0$ when it is true) to $\alpha$. The consistency of this test under $\gamma_k$ (for any fixed $\sigma$) is proved in [6]. A similar result can be shown for $\gamma$ under some conditions on $\mathcal{K}$. We skip the details here.

In our experiments, we set $m = n = 1000$, $\sigma_p^2 = 10$ and draw two sets of independent random samples from $\mathbb{Q}$. The distribution of $T_{mn}$ is estimated by bootstrapping on these samples (250 bootstrap iterations are performed) and the associated $95^{th}$ quantile (we choose $\alpha = 0.05$) is computed. Since the performance of the test is judged by its type-II error (the probability of accepting $H_0$ when $H_1$ is true), we draw a random sample, one each from $\mathbb{P}$ and $\mathbb{Q}$ and test whether $\mathbb{P} = \mathbb{Q}$. This process is repeated 300 times, and estimates of type-I and type-II errors are obtained for both $\gamma$ and $\gamma_k$. 14 different values for $\sigma$ are considered on a logarithmic scale of base 2 with exponents $(-3, -2, -1, 0, 1, \frac{3}{2}, 2, \frac{5}{2}, 3, \frac{7}{2}, 4, 5, 6)$ along with the median distance between samples as one more choice. 5 different choices for $\nu$ are considered: $(\frac{1}{2}, \frac{3}{4}, 1, \frac{5}{4}, \frac{3}{2})$.

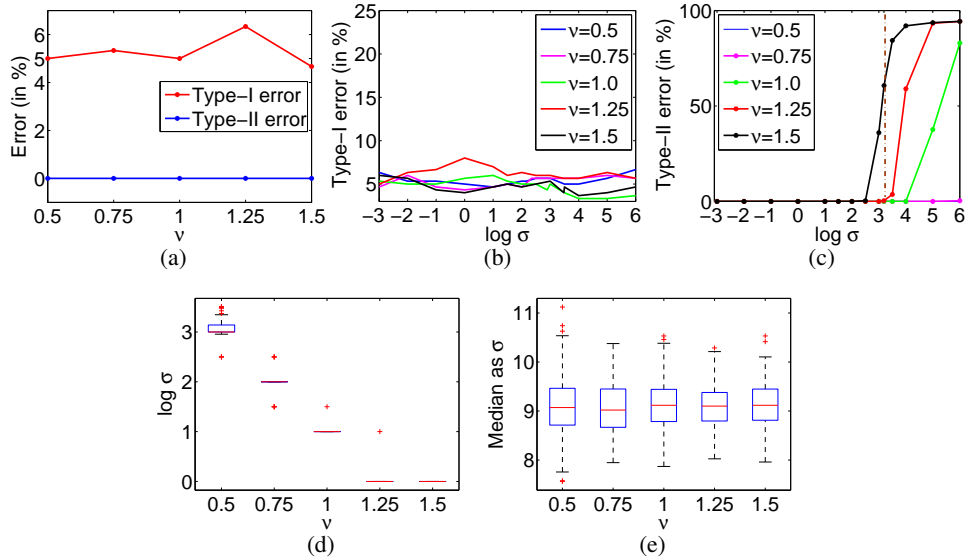

Figure 1: (a) Type-I and Type-II errors (in %) for $\gamma$ for varying $\nu$. (b,c) Type-I and type-II error (in %) for $\gamma_k$ (with different $\sigma$) for varying $\nu$. The dotted line in (c) corresponds to the median heuristic, which shows that its associated type-II error is very large at large $\nu$. (d) Box plot of $\log \sigma$ grouped by $\nu$, where $\sigma$ is selected by $\gamma$. (e) Box plot of the median distance between points (which is also a choice for $\sigma$), grouped by $\nu$. Refer to Section 5 for details.

Figure 1(a) shows the estimated type-I and type-II errors using $\gamma$ as the test statistic for varying $\nu$. Note that the type-I error is close to its design value of $5\%$, while the type-II error is zero for all $\nu$, which means $\gamma$ distinguishes between $\mathbb{P}$ and $\mathbb{Q}$ for all perturbations. Figures 1(b,c) show the estimates of type-I and type-II errors using $\gamma_k$ as the test statistic for different $\sigma$ and $\nu$. Figure 1(d) shows the box plot for $\log \sigma$, grouped by $\nu$, where $\sigma$ is the bandwidth selected by $\gamma$. Figure 1(e) shows the box plot of the median distance between points (which is also a choice for $\sigma$), grouped by $\nu$. From Figures 1(c) and (e), it is easy to see that the median heuristic exhibits high type-II error for $\nu = \frac{3}{2}$, while $\gamma$ exhibits zero type-II error (from Figure 1(a)). Figure 1(c) also shows that heuristic choices of $\sigma$ can result in high type-II errors. It is intuitive to note that as $\nu$ increases, (which means the characteristic function of $\mathbb{Q}$ differs from that of $\mathbb{P}$ at higher frequencies), a smaller $\sigma$ is needed to detect these changes. The advantage of using $\gamma$ is that it selects $\sigma$ in a distribution-dependent fashion and its behavior in the box plot shown in Figure 1(d) matches with the previously mentioned intuition about the behavior of $\sigma$ with respect to $\nu$. These results demonstrate the validity of using $\gamma$ as a distance measure in applications.

## 6 Conclusions

In this work, we have shown how MMD appears in binary classification, and thus that characteristic kernels are important in kernel-based classification algorithms. We have broadened the class of characteristic RKHSs to include those induced by *strictly positive definite* kernels (with particular application to kernels on non-compact domains, and/or kernels that are not translation invariant). We have further provided a convergent generalization of MMD over families of kernel functions, which becomes necessary even in considering relatively simple families of kernels (such as the Gaussian kernels parameterized by their bandwidth). The usefulness of the generalized MMD is illustrated experimentally with a two-sample testing problem.

**Acknowledgments**

The authors thank anonymous reviewers for their constructive comments and especially the reviewer who pointed out the connection between characteristic kernels and the achievability of Bayes risk. B. K. S. was supported by the MPI for Biological Cybernetics, National Science Foundation (grant DMS-MSPA 0625409), the Fair Isaac Corporation and the University of California MICRO program. A. G. was supported by grants DARPA IPTO FA8750-09-1-0141, ONR MURI N000140710747, and ARO MURI W911NF0810242.

## Footnotes

[1]There is a subtlety here, since unlike the problem of testing for differences in distributions, classification suffers from slow learning rates. See [3, Chapter 7] for details.

# References

[1] M. Anthony and P. L. Bartlett. *Neural Network Learning: Theoretical Foundations*. Cambridge University Press, UK, 1999.

[2] V. H. de la Peña and E. Giné. *Decoupling: From Dependence to Independence*. Springer-Verlag, NY, 1999.

[3] L. Devroye, L. Gyorfi, and G. Lugosi. *A Probabilistic Theory of Pattern Recognition*. Springer-Verlag, New York, 1996.

[4] K. Fukumizu, A. Gretton, X. Sun, and B. Schölkopf. Kernel measures of conditional dependence. In J.C. Platt, D. Koller, Y. Singer, and S. Roweis, editors, *Advances in Neural Information Processing Systems 20*, pages 489–496, Cambridge, MA, 2008. MIT Press.

[5] K. Fukumizu, B. K. Sriperumbudur, A. Gretton, and B. Schölkopf. Characteristic kernels on groups and semigroups. In D. Koller, D. Schuurmans, Y. Bengio, and L. Bottou, editors, *Advances in Neural Information Processing Systems 21*, pages 473–480, 2009.

[6] A. Gretton, K. M. Borgwardt, M. Rasch, B. Schölkopf, and A. Smola. A kernel method for the two sample problem. In B. Schölkopf, J. Platt, and T. Hoffman, editors, *Advances in Neural Information Processing Systems 19*, pages 513–520. MIT Press, 2007.

[7] A. Gretton, K. Fukumizu, C.-H. Teo, L. Song, B. Schölkopf, and A. Smola. A kernel statistical test of independence. In *Advances in Neural Information Processing Systems 20*, pages 585–592. MIT Press, 2008.

[8] G. R. G. Lanckriet, N. Christianini, P. Bartlett, L. El Ghaoui, and M. I. Jordan. Learning the kernel matrix with semidefinite programming. *Journal of Machine Learning Research*, 5:24–72, 2004.

[9] B. Schölkopf and A. J. Smola. *Learning with Kernels*. MIT Press, Cambridge, MA, 2002.

[10] B. Schölkopf, B. K. Sriperumbudur, A. Gretton, and K. Fukumizu. RKHS representation of measures. In *Learning Theory and Approximation Workshop*, Oberwolfach, Germany, 2008.

[11] J. Shawe-Taylor and N. Cristianini. *Kernel Methods for Pattern Analysis*. Cambridge University Press, UK, 2004.

[12] A. J. Smola, A. Gretton, L. Song, and B. Schölkopf. A Hilbert space embedding for distributions. In *Proc. 18th International Conference on Algorithmic Learning Theory*, pages 13–31. Springer-Verlag, Berlin, Germany, 2007.

[13] N. Srebro and S. Ben-David. Learning bounds for support vector machines with learned kernels. In G. Lugosi and H. U. Simon, editors, *Proc. of the $19^{th}$ Annual Conference on Learning Theory*, pages 169–183, 2006.

[14] B. K. Sriperumbudur, A. Gretton, K. Fukumizu, G. R. G. Lanckriet, and B. Schölkopf. Injective Hilbert space embeddings of probability measures. In R. Servedio and T. Zhang, editors, *Proc. of the $21^{st}$ Annual Conference on Learning Theory*, pages 111–122, 2008.

[15] I. Steinwart. On the influence of the kernel on the consistency of support vector machines. *Journal of Machine Learning Research*, 2:67–93, 2002.

[16] I. Steinwart and A. Christmann. *Support Vector Machines*. Springer, 2008.

[17] J. Stewart. Positive definite functions and generalizations, an historical survey. *Rocky Mountain Journal of Mathematics*, 6(3):409–433, 1976.

[18] Y. Ying and C. Campbell. Generalization bounds for learning the kernel. In *Proc. of the $22^{nd}$ Annual Conference on Learning Theory*, 2009.

[19] Y. Ying and D. X. Zhou. Learnability of Gaussians with flexible variances. *Journal of Machine Learning Research*, 8:249–276, 2007.

